# Perspectives on Sparse Bayesian Learning

**David Wipf,   Jason Palmer,   and   Bhaskar Rao**
Department of Electrical and Computer Engineering
University of California, San Diego, CA 92092
dwipf,japalmer@ucsd.edu, brao@ece.ucsd.edu

## Abstract

Recently, relevance vector machines (RVM) have been fashioned from a sparse Bayesian learning (SBL) framework to perform supervised learning using a weight prior that encourages sparsity of representation. The methodology incorporates an additional set of hyperparameters governing the prior, one for each weight, and then adopts a specific approximation to the full marginalization over all weights and hyperparameters. Despite its empirical success however, no rigorous motivation for this particular approximation is currently available. To address this issue, we demonstrate that SBL can be recast as the application of a rigorous variational approximation to the full model by expressing the prior in a dual form. This formulation obviates the necessity of assuming any hyperpriors and leads to natural, intuitive explanations of why sparsity is achieved in practice.

## 1   Introduction

In an archetypical regression situation, we are presented with a collection of $N$ regressor/target pairs $\{\boldsymbol{\phi}_i \in \Re^M, t_i \in \Re\}_{i=1}^N$ and the goal is to find a vector of weights $\boldsymbol{w}$ such that, in some sense,

$$t_i \approx \boldsymbol{\phi}_i^T \boldsymbol{w}, \; \forall i \quad \text{or} \quad \boldsymbol{t} \approx \Phi \boldsymbol{w}, \tag{1}$$

where $\boldsymbol{t} \triangleq [t_1, \ldots, t_N]^T$ and $\Phi \triangleq [\boldsymbol{\phi}_1, \ldots, \boldsymbol{\phi}_N]^T \in \Re^{N \times M}$. Ideally, we would like to learn this relationship such that, given a new training vector $\boldsymbol{\phi}_*$, we can make accurate predictions of $\boldsymbol{t}_*$, i.e., we would like to avoid overfitting. In practice, this requires some form of regularization, or a penalty on overly complex models.

Recently, a sparse Bayesian learning (SBL) framework has been derived to find robust solutions to (1) [3, 7]. The key feature of this development is the incorporation of a prior on the weights that encourages sparsity in representation, i.e., few non-zero weights. When $\Phi$ is square and formed from a positive-definite kernel function, we obtain the relevance vector machine (RVM), a Bayesian competitor of SVMs with several significant advantages.

### 1.1   Sparse Bayesian Learning

Given a new regressor vector $\boldsymbol{\phi}_*$, the full Bayesian treatment of (1) involves finding the predictive distribution $p(t_*|\boldsymbol{t})$.[1] We typically compute this distribution by marginalizing

over the model weights, i.e.,

$$p(t_*|\boldsymbol{t}) = \frac{1}{p(\boldsymbol{t})} \int p(t_*|\boldsymbol{w})p(\boldsymbol{w},\boldsymbol{t})d\boldsymbol{w}, \tag{2}$$

where the joint density $p(\boldsymbol{w},\boldsymbol{t}) = p(\boldsymbol{t}|\boldsymbol{w})p(\boldsymbol{w})$ combines all relevant information from the training data (likelihood principle) with our prior beliefs about the model weights. The likelihood term $p(\boldsymbol{t}|\boldsymbol{w})$ is assumed to be Gaussian,

$$p(\boldsymbol{t}|\boldsymbol{w}) = (2\pi\sigma^2)^{-N/2} \exp\left(-\frac{1}{2\sigma^2}\|\boldsymbol{t} - \Phi\boldsymbol{w}\|^2\right), \tag{3}$$

where for now we assume that the noise variance $\sigma^2$ is known. For sparse priors $p(\boldsymbol{w})$ (possibly improper), the required integrations, including the computation of the normalizing term $p(\boldsymbol{t})$, are typically intractable, and we are forced to accept some form of approximation to $p(\boldsymbol{w},\boldsymbol{t})$.

Sparse Bayesian learning addresses this issue by introducing a set of hyperparameters into the specification of the problematic weight prior $p(\boldsymbol{w})$ before adopting a particular approximation. The key assumption is that $p(\boldsymbol{w})$ can be expressed as

$$p(\boldsymbol{w}) = \prod_{i=1}^{M} p(w_i) = \prod_{i=1}^{M} \int p(w_i|\gamma_i)p(\gamma_i)d\gamma_i, \tag{4}$$

where $\boldsymbol{\gamma} = [\gamma_1, \ldots, \gamma_M]^T$ represents a vector of hyperparameters, (one for each weight). The implicit SBL derivation presented in [7] can then be reformulated as follows,

$$
\begin{aligned}
p(t_*|\boldsymbol{t}) &= \frac{1}{p(\boldsymbol{t})} \int p(t_*|\boldsymbol{w})p(\boldsymbol{t}|\boldsymbol{w})p(\boldsymbol{w})d\boldsymbol{w} \\
&= \frac{1}{p(\boldsymbol{t})} \int \int p(t_*|\boldsymbol{w})p(\boldsymbol{t}|\boldsymbol{w})p(\boldsymbol{w}|\boldsymbol{\gamma})p(\boldsymbol{\gamma})d\boldsymbol{w}d\boldsymbol{\gamma}.
\end{aligned}
\tag{5}
$$

Proceeding further, by applying Bayes' rule to this expression, we can exploit the plugin rule [2] via,

$$
\begin{aligned}
p(t_*|\boldsymbol{t}) &= \int \int p(t_*|\boldsymbol{w})p(\boldsymbol{t}|\boldsymbol{w})p(\boldsymbol{w}|\boldsymbol{\gamma})\frac{p(\boldsymbol{\gamma}|\boldsymbol{t})}{p(\boldsymbol{t}|\boldsymbol{\gamma})}d\boldsymbol{w}d\boldsymbol{\gamma} \\
&\approx \int \int p(t_*|\boldsymbol{w})p(\boldsymbol{t}|\boldsymbol{w})p(\boldsymbol{w}|\boldsymbol{\gamma})\frac{\delta(\boldsymbol{\gamma}_{MAP})}{p(\boldsymbol{t}|\boldsymbol{\gamma})}d\boldsymbol{w}d\boldsymbol{\gamma} \\
&= \frac{1}{p(\boldsymbol{t};\boldsymbol{\gamma}_{MAP})} \int p(t_*|\boldsymbol{w})p(\boldsymbol{w},\boldsymbol{t};\boldsymbol{\gamma}_{MAP})d\boldsymbol{w}.
\end{aligned}
\tag{6}
$$

The essential difference from (2) is that we have replaced $p(\boldsymbol{w},\boldsymbol{t})$ with the approximate distribution $p(\boldsymbol{w},\boldsymbol{t};\boldsymbol{\gamma}_{MAP}) = p(\boldsymbol{t}|\boldsymbol{w})p(\boldsymbol{w};\boldsymbol{\gamma}_{MAP})$. Also, the normalizing term becomes $\int p(\boldsymbol{w},\boldsymbol{t};\boldsymbol{\gamma}_{MAP})d\boldsymbol{w}$ and we assume that all required integrations can now be handled in closed form. Of course the question remains, how do we structure this new set of parameters $\boldsymbol{\gamma}$ to accomplish this goal? The answer is that the hyperparameters enter as weight prior variances of the form,

$$p(w_i|\gamma_i) = \mathcal{N}(0,\gamma_i). \tag{7}$$

The hyperpriors are given by,

$$p(\gamma_i^{-1}) \propto \gamma_i^{1-a} \exp(-b/\gamma_i), \tag{8}$$

where $a, b > 0$ are constants. The crux of the actual learning procedure presented in [7] is to find some MAP estimate of $\boldsymbol{\gamma}$ (or more accurately, a function of $\boldsymbol{\gamma}$). In practice, we find that many of the estimated $\gamma_i$'s converge to zero, leading to sparse solutions since the corresponding weights, and therefore columns of $\Phi$, can effectively be pruned from the model. The Gaussian assumptions, both on $p(\boldsymbol{t}|\boldsymbol{w})$ and $p(\boldsymbol{w};\boldsymbol{\gamma})$, then facilitate direct, analytic computation of (6).

### 1.2 Ambiguities in Current SBL Derivation

Modern Bayesian analysis is primarily concerned with finding distributions and locations of significant probability mass, not just modes of distributions, which can be very misleading in many cases [6]. With SBL, the justification for the additional level of sophistication (i.e., the inclusion of hyperparameters) is that the adoption of the plugin rule (i.e., the approximation $p(\boldsymbol{w}, \boldsymbol{t}) \approx p(\boldsymbol{w}, \boldsymbol{t}; \boldsymbol{\gamma}_{MAP})$) is reflective of the true mass, at least sufficiently so for predictive purposes. However, no rigorous motivation for this particular claim is currently available nor is it immediately obvious exactly how the mass of this approximate distribution relates to the true mass.

A more subtle difficulty arises because MAP estimation, and hence the plugin rule, is not invariant under a change in parameterization. Specifically, for an invertible function $f(\cdot)$,

$$[f(\boldsymbol{\gamma})]_{MAP} \neq f(\boldsymbol{\gamma}_{MAP}). \qquad (9)$$

Different transformations lead to different modes and ultimately, different approximations to $p(\boldsymbol{w}, \boldsymbol{t})$ and therefore $p(t_*|\boldsymbol{t})$. So how do we decide which one to use? The canonical form of SBL, and the one that has displayed remarkable success in the literature, does not in fact find a mode of $p(\boldsymbol{\gamma}|\boldsymbol{t})$, but a mode of $p(-\log \boldsymbol{\gamma}|\boldsymbol{t})$. But again, why should this mode necessarily be more reflective of the desired mass than any other?

As already mentioned, SBL often leads to sparse results in practice, namely, the approximation $p(\boldsymbol{w}, \boldsymbol{t}; \boldsymbol{\gamma}_{MAP})$ is typically nonzero only on a small subspace of $M$-dimensional $\boldsymbol{w}$ space. The question remains, however, why should an approximation to the full Bayesian treatment necessarily lead to sparse results in practice?

To address all of these ambiguities, we will herein demonstrate that the sparse Bayesian learning procedure outlined above can be recast as the application of a rigorous variational approximation to the distribution $p(\boldsymbol{w}, \boldsymbol{t})$.[2] This will allow us to quantify the exact relationship between the true mass and the approximate mass of this distribution. In effect, we will demonstrate that SBL is attempting to directly capture significant portions of the probability mass of $p(\boldsymbol{w}, \boldsymbol{t})$, while still allowing us to perform the required integrations. This framework also obviates the necessity of assuming any hyperprior $p(\boldsymbol{\gamma})$ and is independent of the (subjective) parameterization (e.g., $\boldsymbol{\gamma}$ or $-\log \boldsymbol{\gamma}$, etc.). Moreover, this perspective leads to natural, intuitive explanations of why sparsity is observed in practice and why, in general, this need not be the case.

## 2 A Variational Interpretation of Sparse Bayesian Learning

To begin, we review that the ultimate goal of this analysis is to find a well-motivated approximation to the distribution

$$p(t_*|\boldsymbol{t}; \mathcal{H}) \propto \int p(t_*|\boldsymbol{w}) p(\boldsymbol{w}, \boldsymbol{t}; \mathcal{H}) d\boldsymbol{w} = \int p(t_*|\boldsymbol{w}) p(\boldsymbol{t}|\boldsymbol{w}) p(\boldsymbol{w}; \mathcal{H}) d\boldsymbol{w}, \qquad (10)$$

where we have explicitly noted the hypothesis of a model with a sparsity inducing (possibly improper) weight prior by $\mathcal{H}$. As already mentioned, the integration required by this form is analytically intractable and we must resort to some form of approximation. To accomplish this, we appeal to variational methods to find a viable approximation to $p(\boldsymbol{w}, \boldsymbol{t}; \mathcal{H})$ [5]. We may then substitute this approximation into (10), leading to tractable integrations and analytic posterior distributions. To find a class of suitable approximations, we first express $p(\boldsymbol{w}; \mathcal{H})$ in its dual form by introducing a set of variational parameters. This is similar to a procedure outlined in [4] in the context of independent component analysis.

### 2.1 Dual Form Representation of $p(\boldsymbol{w}; \mathcal{H})$

At the heart of this methodology is the ability to represent a convex function in its dual form. For example, given a convex function $f(y) : \Re \to \Re$, the dual form is given by

$$f(y) = \sup_\lambda \left[ \lambda y - f^*(\lambda) \right], \tag{11}$$

where $f^*(\lambda)$ denotes the conjugate function. Geometrically, this can be interpreted as representing $f(x)$ as the upper envelope or supremum of a set of lines parameterized by $\lambda$. The selection of $f^*(\lambda)$ as the intercept term ensures that each line is tangent to $f(y)$. If we drop the maximization in (11), we obtain the bound

$$f(y) \geq \lambda y - f^*(\lambda). \tag{12}$$

Thus, for any given $\lambda$, we have a lower bound on $f(y)$; we may then optimize over $\lambda$ to find the optimal or tightest bound in a region of interest.

To apply this theory to the problem at hand, we specify the form for our sparse prior $p(\boldsymbol{w}; \mathcal{H}) = \prod_{i=1}^{M} p(w_i; \mathcal{H})$. Using (7) and (8), we obtain the prior

$$p(w_i; \mathcal{H}) = \int p(w_i|\gamma_i) p(\gamma_i) d\gamma_i = C \left( b + \frac{w_i^2}{2} \right)^{-(a+1/2)}, \tag{13}$$

which for $a, b > 0$ is proportional to a Student-$t$ density. The constant $C$ is not chosen to enforce proper normalization; rather, it is chosen to facilitate the variational analysis below. Also, this density function can be seen to encourage sparsity since it has heavy tails and a sharp peak at zero. Clearly $p(w_i; \mathcal{H})$ is not convex in $w_i$; however, if we let $y_i \triangleq w_i^2$ as suggested in [5] and define

$$f(y_i) \triangleq \log p(w_i; \mathcal{H}) = -(a + 1/2) \log C \left( b + \frac{y_i}{2} \right), \tag{14}$$

we see that we now have a convex function in $y_i$ amenable to dual representation. By computing the conjugate function $f^*(y_i)$, constructing the dual, and then transforming back to $p(w_i; \mathcal{H})$, we obtain the representation (see Appendix for details)

$$p(w_i; \mathcal{H}) = \max_{\gamma_i \geq 0} \left[ (2\pi\gamma_i)^{-1/2} \exp \left( -\frac{w_i^2}{2\gamma_i} \right) \exp \left( -\frac{b}{\gamma_i} \right) \gamma_i^{-a} \right]. \tag{15}$$

As $a, b \to 0$, it is readily apparent from (15) that what were straight lines in the $y_i$ domain are now Gaussian functions with variance $\gamma_i$ in the $w_i$ domain. Figure 1 illustrates this connection. When we drop the maximization, we obtain a lower bound on $p(w_i; \mathcal{H})$ of the form

$$p(w_i; \mathcal{H}) \geq p(w_i; \hat{\mathcal{H}}) \triangleq (2\pi\gamma_i)^{-1/2} \exp \left( -\frac{w_i^2}{2\gamma_i} \right) \exp \left( -\frac{b}{\gamma_i} \right) \gamma_i^{-a}, \tag{16}$$

which serves as our approximate prior to $p(\boldsymbol{w}; \mathcal{H})$. From this relationship, we see that $p(w_i; \hat{\mathcal{H}})$ does not integrate to one, except in the special case when $a, b \to 0$. We will now incorporate these results into an algorithm for finding a good $\hat{\mathcal{H}}$, or more accurately $\hat{\mathcal{H}}(\boldsymbol{\gamma})$, since each candidate hypothesis is characterized by a different set of variational parameters.

### 2.2 Variational Approximation to $p(\boldsymbol{w}, \boldsymbol{t}; \mathcal{H})$

So now that we have a variational approximation to the problematic weight prior, we must return to our original problem of estimating $p(t_*|\boldsymbol{t}; \mathcal{H})$. Since the integration is intractable under model hypothesis $\mathcal{H}$, we will instead compute $p(t_*|\boldsymbol{t}; \hat{\mathcal{H}})$ using $p(\boldsymbol{w}, \boldsymbol{t}; \hat{\mathcal{H}}) = p(\boldsymbol{t}|\boldsymbol{w}) p(\boldsymbol{w}; \hat{\mathcal{H}})$, with $p(\boldsymbol{w}; \hat{\mathcal{H}})$ defined as in (16). How do we choose this approximate

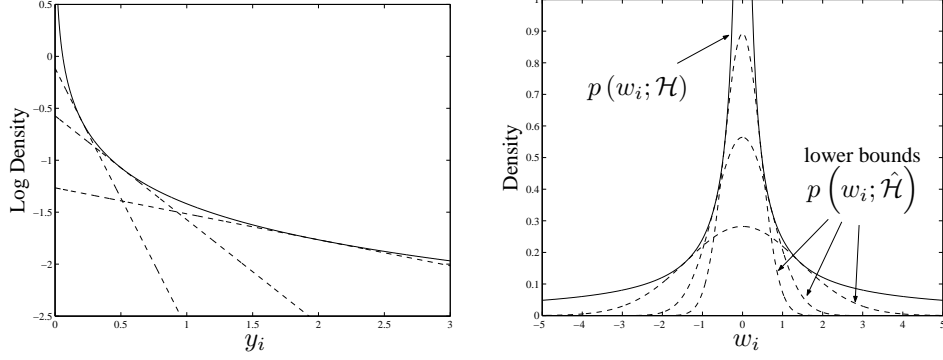

Figure 1: Variational approximation example in both $y_i$ space and $w_i$ space for $a, b \to 0$. *Left*: Dual forms in $y_i$ space. The solid line represents the plot of $f(y_i)$ while the dotted lines represent variational lower bounds in the dual representation for three different values of $\lambda_i$. *Right*: Dual forms in $w_i$ space. The solid line represents the plot of $p(w_i; \mathcal{H})$ while the dotted lines represent Gaussian distributions with three different variances.

model? In other words, given that different $\hat{\mathcal{H}}$ are distinguished by a different set of variational parameters $\boldsymbol{\gamma}$, how do we choose the most appropriate $\boldsymbol{\gamma}$? Consistent with modern Bayesian analysis, we concern ourselves not with matching modes of distributions, but with aligning regions of significant probability mass. In choosing $p(\boldsymbol{w}, \boldsymbol{t}; \hat{\mathcal{H}})$, we would therefore like to match, where possible, significant regions of probability mass in the true model $p(\boldsymbol{w}, \boldsymbol{t}; \mathcal{H})$. For a given $\boldsymbol{t}$, an obvious way to do this is to select $\hat{\mathcal{H}}$ by minimizing the sum of the misaligned mass, i.e.,

$$
\begin{aligned}
\hat{\mathcal{H}} &= \arg\min_{\hat{\mathcal{H}}} \int \left| p(\boldsymbol{w}, \boldsymbol{t}; \mathcal{H}) - p(\boldsymbol{w}, \boldsymbol{t}; \hat{\mathcal{H}}) \right| d\boldsymbol{w} \\
&= \arg\max_{\hat{\mathcal{H}}} \int p(\boldsymbol{t}|\boldsymbol{w}) p(\boldsymbol{w}; \hat{\mathcal{H}}) d\boldsymbol{w},
\end{aligned}
\tag{17}
$$

where the variational assumptions have allowed us to remove the absolute value (since the argument must always be positive). Also, we note that (17) is tantamount to selecting the variational approximation with maximal Bayesian evidence [6]. In other words, we are selecting the $\hat{\mathcal{H}}$, out of a class of variational approximations to $\mathcal{H}$, that most probably explains the training data $\boldsymbol{t}$, marginalized over the weights.

From an implementational standpoint, (17) can be reexpressed using (16) as,

$$
\begin{aligned}
\boldsymbol{\gamma} &= \arg\max_{\boldsymbol{\gamma}} \log \int p(\boldsymbol{t}|\boldsymbol{w}) \prod_{i=1}^{M} p\left(w_i; \hat{\mathcal{H}}(\gamma_i)\right) d\boldsymbol{w} \\
&= \arg\max_{\boldsymbol{\gamma}} -\frac{1}{2}\left[\log|\Sigma_t| + \boldsymbol{t}^T \Sigma_t^{-1} \boldsymbol{t}\right] + \sum_{i=1}^{M}\left(-\frac{b}{\gamma_i} - a\log\gamma_i\right),
\end{aligned}
\tag{18}
$$

where $\Sigma_t \triangleq \sigma^2 I + \Phi\mathrm{diag}(\boldsymbol{\gamma})\Phi^T$. This is the same cost function as in [7] only without terms resulting from a prior on $\sigma^2$, which we will address later. Thus, the end result of this analysis is an evidence maximization procedure equivalent to the one in [7]. The difference is that, where before we were optimizing over a somewhat arbitrary model parameterization, now we see that it is actually optimization over the space of variational approximations to a model with a sparse, regularizing prior. Also, we know from (17) that this procedure is effectively matching, as much as possible, the mass of the full model $p(\boldsymbol{w}, \boldsymbol{t}; \hat{\mathcal{H}})$.

## 3 Analysis

While the variational perspective is interesting, two pertinent questions still remain:

1. Why should it be that approximating a sparse prior $p(\boldsymbol{w}; \mathcal{H})$ leads to sparse representations in practice?
2. How do we extend these results to handle an unknown, random variance $\sigma^2$?

We first treat *Question (1)*. In Figure 2 below, we have illustrated a $2D$ example of evidence maximization within the context of variational approximations to the sparse prior $p(\boldsymbol{w}; \mathcal{H})$. For now, we will assume $a, b \to 0$, which from (13), implies that $p(w_i; \mathcal{H}) \propto 1/|w_i|$ for each $i$. On the left, the shaded area represents the region of $\boldsymbol{w}$ space where both $p(\boldsymbol{w}; \mathcal{H})$ and $p(\boldsymbol{t}|\boldsymbol{w})$ (and therefore $p(\boldsymbol{w}, \boldsymbol{t}; \mathcal{H})$) have significant probability mass. Maximization of (17) involves finding an approximate distribution $p(\boldsymbol{w}, \boldsymbol{t}; \hat{\mathcal{H}})$ with a substantial percentage of its mass in this region.

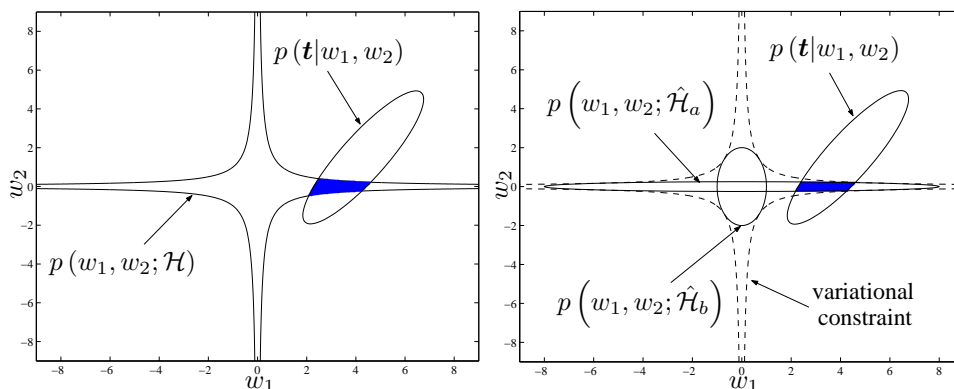

Figure 2: Comparison between full model and approximate models with $a, b \to 0$. *Left:* Contours of equiprobability density for $p(\boldsymbol{w}; \mathcal{H})$ and constant likelihood $p(\boldsymbol{t}|\boldsymbol{w})$; the prominent density and likelihood lie within each region respectively. The shaded region represents the area where both have significant mass. *Right:* Here we have added the contours of $p(\boldsymbol{w}; \hat{\mathcal{H}})$ for two different values of $\boldsymbol{\gamma}$, i.e., two approximate hypotheses denoted $\hat{\mathcal{H}}_a$ and $\hat{\mathcal{H}}_b$. The shaded region represents the area where both the likelihood and the *approximate* prior $\hat{\mathcal{H}}_a$ have significant mass. Note that by the variational bound, each $p(\boldsymbol{w}; \hat{\mathcal{H}})$ must lie within the contours of $p(\boldsymbol{w}; \mathcal{H})$.

In the plot on the right, we have graphed two approximate priors that satisfy the variational bounds, i.e., they must lie within the contours of $p(\boldsymbol{w}; \mathcal{H})$. We see that the narrow prior that aligns with the horizontal spine of $p(\boldsymbol{w}; \mathcal{H})$ places the largest percentage of its mass (and therefore the mass of $p(\boldsymbol{w}, \boldsymbol{t}; \hat{\mathcal{H}}_a)$) in the shaded region. This corresponds with a prior of

$$p(\boldsymbol{w}; \hat{\mathcal{H}}_a) = p(w_1, w_2; \gamma_1 \gg 0, \gamma_2 \approx 0). \tag{19}$$

This creates a long narrow prior since there is minimal variance along the $w_2$ axis. In fact, it can be shown that owing to the infinite density of the variational constraint along each axis (which is allowed as $a$ and $b$ go to zero), the maximum evidence is obtained when $\gamma_2$ is strictly equal to zero, giving the approximate prior infinite density along this axis as well. This implies that $w_2$ also equals zero and can be pruned from the model. In contrast, a model with significant prior variance along both axes, $\hat{\mathcal{H}}_b$, is hampered because it cannot extend directly out (due to the dotted variational boundary) along the spine to penetrate the likelihood.

Similar effective weight pruning occurs in higher dimensional problems as evidenced by simulation studies and the analysis in [3]. In higher dimensions, the algorithm only retains those weights associated with the prior spines that span a subspace penetrating the most prominent portion of the likelihood mass (i.e., a higher-dimensional analog to the shaded region already mentioned). The prior $p(\boldsymbol{w}; \hat{\mathcal{H}})$ navigates the variational constraints, placing as much as possible of its mass in this region, driving many of the $\gamma_i$'s to zero.

In contrast, when $a, b > 0$, the situation is somewhat different. It is not difficult to show that, assuming a noise variance $\sigma^2 > 0$, the variational approximation to $p(\boldsymbol{w}, \boldsymbol{t}; \mathcal{H})$ with maximal evidence cannot have any $\gamma_i = w_i = 0$. Intuitively, this occurs because the now *finite* spines of the prior $p(\boldsymbol{w}; \mathcal{H})$, which bound the variational approximation, do not allow us to place infinite prior density in any region of weight space (as occurred previously when any $\gamma_i \rightarrow 0$). Consequently, if any $\gamma_i$ goes to zero with $a, b > 0$, the associated approximate prior mass, and therefore the approximate evidence, must also fall to zero by (16). As such, *models with all non-zero weights will be now be favored when we form the variational approximation*. We therefore cannot assume an approximation to a sparse prior will necessarily give us sparse results in practice.

We now address *Question (2)*. Thus far, we have considered a known, fixed noise variance $\sigma^2$; however, what if $\sigma^2$ is unknown? SBL assumes it is unknown and random with prior distribution $p(1/\sigma^2) \propto (\sigma^2)^{1-c} \exp(-d/\sigma^2)$, and $c, d > 0$. After integrating out the unknown $\sigma^2$, we arrive at the implicit likelihood equation,

$$p(\boldsymbol{t}|\boldsymbol{w}) = \int p(\boldsymbol{t}|\boldsymbol{w}, \sigma^2)p(\sigma^2)d\sigma^2 \propto \left(d + \frac{1}{2}\|\boldsymbol{t} - \Phi\boldsymbol{w}\|^2\right)^{-(\bar{c}+1/2)}, \qquad (20)$$

where $\bar{c} \triangleq c + (N-1)/2$. We may then form a variational approximation to the likelihood in a similar manner as before (with $w_i$ being replaced by $\|\boldsymbol{t} - \Phi\boldsymbol{w}\|$) giving us,

$$
\begin{aligned}
p(\boldsymbol{t}|\boldsymbol{w}) &\geq (2\pi)^{-N/2}(\sigma^2)^{-1/2} \exp\left(-\frac{1}{2\sigma^2}\|\boldsymbol{t} - \Phi\boldsymbol{w}\|^2\right) \exp\left(-\frac{d}{\sigma^2}\right)(\sigma^2)^{-\bar{c}} \\
&= (2\pi\sigma^2)^{-N/2} \exp\left(-\frac{1}{2\sigma^2}\|\boldsymbol{t} - \Phi\boldsymbol{w}\|^2\right) \exp\left(-\frac{d}{\sigma^2}\right)(\sigma^2)^{-c}, \qquad (21)
\end{aligned}
$$

where the second step follows by substituting back in for $\bar{c}$. By replacing $p(\boldsymbol{t}|\boldsymbol{w})$ with the lower bound from (21), we then maximize over the variational parameters $\boldsymbol{\gamma}$ and $\sigma^2$ via

$$\boldsymbol{\gamma}, \sigma^2 = \arg\max_{\boldsymbol{\gamma}, \sigma^2} -\frac{1}{2}\left[\log|\Sigma_t| + \boldsymbol{t}^T\Sigma_t^{-1}\boldsymbol{t}\right] + \sum_{i=1}^{M}\left(-\frac{b}{\gamma_i} - a\log\gamma_i\right) - \frac{d}{\sigma^2} - c\log\sigma^2, \ (22)$$

the exact SBL optimization procedure. Thus, we see that the entire SBL framework, including noise variance estimation, can be seen in variational terms.

## 4 Conclusions

The end result of this analysis is an evidence maximization procedure that is equivalent to the one originally formulated in [7]. The difference is that, where before we were optimizing over a somewhat arbitrary model parameterization, we now see that SBL is actually searching a space of variational approximations to find an alternative distribution that captures the significant mass of the full model. Moreover, from the vantage point afforded by this new perspective, we can better understand the sparsity properties of SBL and the relationship between sparse priors and approximations to sparse priors.

# Appendix: Derivation of the Dual Form of $p(w_i; \mathcal{H})$

To accommodate the variational analysis of Sec. 2.1, we require the dual representation of $p(w_i; \mathcal{H})$. As an intermediate step, we must find the dual representation of $f(y_i)$, where $y_i \triangleq w_i^2$ and

$$f(y_i) \triangleq \log p(w_i; \mathcal{H}) = \log \left[ C \left( b + \frac{y_i}{2} \right)^{-(a+1/2)} \right]. \tag{23}$$

To accomplish this, we find the conjugate function $f^*(\lambda_i)$ using the duality relation

$$f^*(\lambda_i) = \max_{y_i} \left[ \lambda_i y_i - f(y_i) \right] = \max_{y_i} \left[ \lambda_i y_i - \log C + \left( a + \frac{1}{2} \right) \log \left( b + \frac{y_i}{2} \right) \right]. \tag{24}$$

To find the maximizing $y_i$, we take the gradient of the left side and set it to zero, giving us,

$$y_i^{max} = -\frac{a}{\lambda_i} - \frac{1}{2\lambda_i} - 2b. \tag{25}$$

Substituting this value into the expression for $f^*(\lambda_i)$ and selecting

$$C = (2\pi)^{-1/2} \exp \left[ -\left( a + \frac{1}{2} \right) \right] \left( a + \frac{1}{2} \right)^{(a+1/2)}, \tag{26}$$

we arrive at

$$f^*(\lambda_i) = \left( a + \frac{1}{2} \right) \log \left( \frac{-1}{2\lambda_i} \right) + \frac{1}{2} \log 2\pi - 2b\lambda_i. \tag{27}$$

We are now ready to represent $f(y_i)$ in its dual form, observing first that we only need consider maximization over $\lambda_i \leq 0$ since $f(y_i)$ is a monotonically decreasing function (i.e., all tangent lines will have negative slope). Proceeding forward, we have

$$f(y_i) = \max_{\lambda_i \leq 0} \left[ \lambda_i y_i - f^*(\lambda_i) \right] = \max_{\gamma_i \geq 0} \left[ \frac{-y_i}{2\gamma_i} - \left( a + \frac{1}{2} \right) \log \gamma_i - \frac{1}{2} \log 2\pi - \frac{b}{\gamma_i} \right], \tag{28}$$

where we have used the monotonically increasing transformation $\lambda_i = -1/(2\gamma_i), \gamma_i \geq 0$. The attendant dual representation of $p(w_i; \mathcal{H})$ can then be obtained by exponentiating both sides of (28) and substituting $y_i = w_i^2$,

$$p(w_i; \mathcal{H}) = \max_{\gamma_i \geq 0} \left[ \frac{1}{\sqrt{2\pi\gamma_i}} \exp \left( -\frac{w_i^2}{2\gamma_i} \right) \exp \left( -\frac{b}{\gamma_i} \right) \gamma_i^{-a} \right]. \tag{29}$$

## Acknowledgments

This research was supported by DiMI grant #22-8376 sponsored by Nissan.

## Footnotes

[1] For simplicity, we omit explicit conditioning on $\Phi$ and $\boldsymbol{\phi}^*$, i.e., $p(t^*|\boldsymbol{t}) \equiv p(t^*|\boldsymbol{t}, \Phi, \boldsymbol{\phi}^*)$.

[2]We note that the analysis in this paper is different from [1], which derives an alternative SBL algorithm based on variational methods.

## References

[1] C. Bishop and M. Tipping, "Variational relevance vector machines," *Proc. 16th Conf. Uncertainty in Artificial Intelligence*, pp. 46–53, 2000.

[2] R. Duda, P. Hart, and D. Stork, *Pattern Classification*, Wiley, Inc., New York, 2nd ed., 2001.

[3] A.C. Faul and M.E. Tipping, "Analysis of sparse Bayesian learning," *Advances in Neural Information Processing Systems 14*, pp. 383–389, 2002.

[4] M. Girolami, "A variational method for learning sparse and overcomplete representations," *Neural Computation*, vol. 13, no. 11, pp. 2517–2532, 2001.

[5] M.I. Jordan, Z. Ghahramani, T. Jaakkola, and L.K. Saul, "An introduction to variational methods for graphical models," *Machine Learning*, vol. 37, no. 2, pp. 183–233, 1999.

[6] D.J.C. MacKay, "Bayesian interpolation," *Neural Comp.*, vol. 4, no. 3, pp. 415–447, 1992.

[7] M.E. Tipping, "Sparse Bayesian learning and the relevance vector machine," *Journal of Machine Learning*, vol. 1, pp. 211–244, 2001.
